# Estimating divergence functionals and the likelihood ratio by penalized convex risk minimization

**XuanLong Nguyen**
SAMSI & Duke University

**Martin J. Wainwright**
UC Berkeley

**Michael I. Jordan**
UC Berkeley

## Abstract

We develop and analyze an algorithm for nonparametric estimation of divergence functionals and the density ratio of two probability distributions. Our method is based on a variational characterization of $f$-divergences, which turns the estimation into a penalized convex risk minimization problem. We present a derivation of our kernel-based estimation algorithm and an analysis of convergence rates for the estimator. Our simulation results demonstrate the convergence behavior of the method, which compares favorably with existing methods in the literature.

## 1 Introduction

An important class of "distances" between multivariate probability distributions $\mathbb{P}$ and $\mathbb{Q}$ are the Ali-Silvey or $f$-divergences [1, 6]. These divergences, to be defined formally in the sequel, are all of the form $D_\phi(\mathbb{P}, \mathbb{Q}) = \int \phi(d\mathbb{Q}/d\mathbb{P})d\mathbb{P}$, where $\phi$ is a convex function of the likelihood ratio. This family, including the Kullback-Leibler (KL) divergence and the variational distance as special cases, plays an important role in various learning problems, including classification, dimensionality reduction, feature selection and independent component analysis. For all of these problems, if $f$-divergences are to be used as criteria of merit, one has to be able to estimate them efficiently from data.

With this motivation, the focus of paper is the problem of estimating an $f$-divergence based on i.i.d. samples from each of the distributions $\mathbb{P}$ and $\mathbb{Q}$. Our starting point is a variational characterization of $f$-divergences, which allows our problem to be tackled via an $M$-estimation procedure. Specifically, the likelihood ratio function $d\mathbb{P}/d\mathbb{Q}$ and the divergence functional $D_\phi(\mathbb{P}, \mathbb{Q})$ can be estimated by solving a convex minimization problem over a function class. In this paper, we estimate the likelihood ratio and the KL divergence by optimizing a *penalized* convex risk. In particular, we restrict the estimate to a bounded subset of a reproducing kernel Hilbert Space (RKHS) [17]. The RKHS is sufficiently rich for many applications, and also allows for computationally efficient optimization procedures. The resulting estimator is nonparametric, in that it entails no strong assumptions on the form of $\mathbb{P}$ and $\mathbb{Q}$, except that the likelihood ratio function is assumed to belong to the RKHS.

The bulk of this paper is devoted to the derivation of the algorithm, and a theoretical analysis of the performance of our estimator. The key to our analysis is a basic inequality relating a performance metric (the Hellinger distance) of our estimator to the suprema of two empirical processes (with respect to $\mathbb{P}$ and $\mathbb{Q}$) defined on a function class of density ratios. Convergence rates are then obtained using techniques for analyzing nonparametric $M$-estimators from empirical process theory [20].

**Related work.** The variational representation of divergences has been derived independently and exploited by several authors [5, 11, 14]. Broniatowski and Keziou [5] studied testing and estimation problems based on dual representations of $f$-divergences, but working in a parametric setting as opposed to the nonparametric framework considered here. Nguyen et al. [14] established a one-to-one correspondence between the family of $f$-divergences and the family of surrogate loss functions [2], through which the (optimum) "surrogate risk" is equal to the negative of an associated $f$-divergence. Another link is to the problem of estimating integral functionals of a single density, with the Shannon entropy being a well-known example, which has been studied extensively dating back to early

work [9, 13] as well as the more recent work [3, 4, 12]. See also [7, 10, 8] for the problem of (Shannon) entropy functional estimation. In another branch of related work, Wang et al. [22] proposed an algorithm for estimating the KL divergence for continuous distributions, which exploits histogram-based estimation of the likelihood ratio by building data-dependent partitions of equivalent (empirical) $\mathbb{Q}$-measure. The estimator was empirically shown to outperform direct plug-in methods, but no theoretical results on its convergence rate were provided.

This paper is organized as follows. Sec. 2 provides a background of $f$-divergences. In Sec. 3, we describe an estimation procedure based on penalized risk minimization and accompanying convergence rates analysis results. In Sec. 4, we derive and implement efficient algorithms for solving these problems using RKHS. Sec. 5 outlines the proof of the analysis. In Sec. 6, we illustrate the behavior of our estimator and compare it to other methods via simulations.

## 2 Background

We begin by defining $f$-divergences, and then provide a variational representation of the $f$-divergence, which we later exploit to develop an $M$-estimator.

Consider two distributions $\mathbb{P}$ and $\mathbb{Q}$, both assumed to be absolutely continuous with respect to Lebesgue measure $\mu$, with positive densities $p_0$ and $q_0$, respectively, on some compact domain $\mathcal{X} \subset \mathbb{R}^d$. The class of Ali-Silvey or $f$-divergences [6, 1] are "distances" of the form:

$$D_\phi(\mathbb{P}, \mathbb{Q}) \quad = \quad \int p_0 \phi(q_0/p_0) \, d\mu, \tag{1}$$

where $\phi : \mathbb{R} \rightarrow \bar{\mathbb{R}}$ is a convex function. Different choices of $\phi$ result in many divergences that play important roles in information theory and statistics, including the variational distance, Hellinger distance, KL divergence and so on (see, e.g., [19]). As an important example, the Kullback-Leibler (KL) divergence between $\mathbb{P}$ and $\mathbb{Q}$ is given by $D_K(\mathbb{P}, \mathbb{Q}) = \int p_0 \log(p_0/q_0) \, d\mu$, corresponding to the choice $\phi(t) = -\log(t)$ for $t > 0$ and $+\infty$ otherwise.

**Variational representation:** Since $\phi$ is a convex function, by Legendre-Fenchel convex duality [16] we can write $\phi(u) = \sup_{v \in \mathbb{R}} (uv - \phi^*(v))$, where $\phi^*$ is the convex conjugate of $\phi$. As a result,

$$D_\phi(\mathbb{P}, \mathbb{Q}) \quad = \quad \int p_0 \sup_f (f q_0/p_0 - \phi^*(f)) \, d\mu \quad = \quad \sup_f \left( \int f \, d\mathbb{Q} - \int \phi^*(f) \, d\mathbb{P} \right),$$

where the supremum is taken over all measurable functions $f : \mathcal{X} \rightarrow \mathbb{R}$, and $\int f \, d\mathbb{P}$ denotes the expectation of $f$ under distribution $\mathbb{P}$. Denoting by $\partial \phi$ the subdifferential [16] of the convex function $\phi$, it can be shown that the supremum will be achieved for functions $f$ such that $q_0/p_0 \in \partial \phi^*(f)$, where $q_0, p_0$ and $f$ are evaluated at any $x \in \mathcal{X}$. By convex duality [16], this is true if $f \in \partial \phi(q_0/p_0)$ for any $x \in \mathcal{X}$. Thus, we have proved [15, 11]:

**Lemma 1.** *Letting $\mathcal{F}$ be any function class in $\mathcal{X} \rightarrow \mathbb{R}$, there holds:*

$$D_\phi(\mathbb{P}, \mathbb{Q}) \geq \sup_{f \in \mathcal{F}} \int f \, d\mathbb{Q} - \phi^*(f) \, d\mathbb{P}, \tag{2}$$

*with equality if $\mathcal{F} \cap \partial \phi(q_0/p_0) \neq \emptyset$.*

To illustrate this result in the special case of the KL divergence, here the function $\phi$ has the form $\phi(u) = -\log(u)$ for $u > 0$ and $+\infty$ for $u \leq 0$. The convex dual of $\phi$ is $\phi^*(v) = \sup_u (uv - \phi(u)) = -1 - \log(-v)$ if $u < 0$ and $+\infty$ otherwise. By Lemma 1,

$$D_K(\mathbb{P}, \mathbb{Q}) \quad = \quad \sup_{f < 0} \int f \, d\mathbb{Q} - \int (-1 - \log(-f)) \, d\mathbb{P} \quad = \quad \sup_{g > 0} \int \log g \, d\mathbb{P} - \int g \, d\mathbb{Q} + 1. \tag{3}$$

In addition, the supremum is attained at $g = p_0/q_0$.

## 3 Penalized M-estimation of KL divergence and the density ratio

Let $X_1, \dots, X_n$ be a collection of $n$ i.i.d. samples from the distribution $\mathbb{Q}$, and let $Y_1, \dots, Y_n$ be $n$ i.i.d. samples drawn from the distribution $\mathbb{P}$. Our goal is to develop an estimator of the KL divergence and the density ratio $g_0 = p_0/q_0$ based on the samples $\{X_i\}_{i=1}^n$ and $\{Y_i\}_{i=1}^n$.

The variational representation in Lemma 1 motivates the following estimator of the KL divergence. First, let $\mathcal{G}$ be a function class of $\mathcal{X} \to \mathbb{R}_+$. We then compute

$$\hat{D}_K = \sup_{g \in \mathcal{G}} \int \log g \, d\mathbb{P}_n - \int g d\mathbb{Q}_n + 1, \tag{4}$$

where $\int \, d\mathbb{P}_n$ and $\int \, d\mathbb{Q}_n$ denote the expectation under empirical measures $\mathbb{P}_n$ and $\mathbb{Q}_n$, respectively. If the supremum is attained at $\hat{g}_n$, then $\hat{g}_n$ serves as an estimator of the density ratio $g_0 = p_0/q_0$.

In practice, the "true" size of $\mathcal{G}$ is not known. Accordingly, our approach in this paper is an alternative approach based on controlling the size of $\mathcal{G}$ by using penalties. More precisely, let $I(g)$ be a non-negative measure of complexity for $g$ such that $I(g_0) < \infty$. We decompose the function class $\mathcal{G}$ as follows:

$$\mathcal{G} = \cup_{1 \le M \le \infty} \mathcal{G}_M, \tag{5}$$

where $\mathcal{G}_M := \{g \mid I(g) \le M\}$ is a ball determined by $I(\cdot)$.

The estimation procedure involves solving the following program:

$$\hat{g}_n = \mathrm{argmin}_{g \in \mathcal{G}} \int g d\mathbb{Q}_n - \int \log g \, d\mathbb{P}_n + \frac{\lambda_n}{2} I^2(g), \tag{6}$$

where $\lambda_n > 0$ is a regularization parameter. The minimizing argument $\hat{g}_n$ is plugged into (4) to obtain an estimate of the KL divergence $D_K$.

For the KL divergence, the difference $|\hat{D}_K - D_K(\mathbb{P}, \mathbb{Q})|$ is a natural performance measure. For estimating the density ratio, various metrics are possible. Viewing $g_0 = p_0/q_0$ as a density function with respect to $\mathbb{Q}$ measure, one useful metric is the (generalized) Hellinger distance:

$$h_{\mathbb{Q}}^2(g_0, g) := \frac{1}{2} \int (g_0^{1/2} - g^{1/2})^2 \, d\mathbb{Q}. \tag{7}$$

For the analysis, several assumptions are in order. First, assume that $g_0$ (*not* all of $\mathcal{G}$) is bounded from above and below:

$$0 < \eta_0 \le g_0 \le \eta_1 \text{ for some constants } \eta_0, \eta_1. \tag{8}$$

Next, the uniform norm of $\mathcal{G}_M$ is Lipchitz with respect to the penalty measure $I(g)$, i.e.:

$$\sup_{g \in \mathcal{G}_M} |g|_\infty \le cM \text{ for any } M \ge 1. \tag{9}$$

Finally, on the bracket entropy of $\mathcal{G}$ [21]: For some $0 < \gamma < 2$,

$$\mathcal{H}_\delta^B(\mathcal{G}_M, L_2(\mathbb{Q})) = O(M/\delta)^\gamma \text{ for any } \delta > 0. \tag{10}$$

The following is our main theoretical result, whose proof is given in Section 5:

**Theorem 2.** *(a) Under assumptions* (8), (9) *and* (10)*, and letting* $\lambda_n \to 0$ *so that:*

$$\lambda_n^{-1} = O_{\mathbb{P}}(n^{2/(2+\gamma)})(1 + I(g_0)),$$

*then under* $\mathbb{P}$*:*

$$h_{\mathbb{Q}}(g_0, \hat{g}_n) = O_{\mathbb{P}}(\lambda_n^{1/2})(1 + I(g_0)), \quad I(\hat{g}_n) = O_{\mathbb{P}}(1 + I(g_0)).$$

*(b) If, in addition to* (8), (9) *and* (10)*, there holds* $\inf_{g \in \mathcal{G}} g(x) \ge \eta_0$ *for any* $x \in \mathcal{X}$*, then*

$$|\hat{D}_K - D_K(\mathbb{P}, \mathbb{Q})| = O_{\mathbb{P}}(\lambda_n^{1/2})(1 + I(g_0)). \tag{11}$$

## 4 Algorithm: Optimization and dual formulation

$\mathcal{G}$ **is an RKHS.** Our algorithm involves solving program (6), for some choice of function class $\mathcal{G}$. In our implementation, relevant function classes are taken to be a reproducing kernel Hilbert space induced by a Gaussian kernel. The RKHS's are chosen because they are sufficiently rich [17], and as in many learning tasks they are quite amenable to efficient optimization procedures [18].

Let $K : \mathcal{X} \times \mathcal{X} \to \mathbb{R}$ be a Mercer kernel function [17]. Thus, $K$ is associated with a feature map $\Phi : \mathcal{X} \to \mathcal{H}$, where $\mathcal{H}$ is a Hilbert space with inner product $\langle ., .\rangle$ and for all $x, x' \in \mathcal{X}$, $K(x, x') = \langle \Phi(x), \Phi(x') \rangle$. As a reproducing kernel Hilbert space, any function $g \in \mathcal{H}$ can be expressed as an inner product $g(x) = \langle w, \Phi(x) \rangle$, where $\|g\|_{\mathcal{H}} = \|w\|_{\mathcal{H}}$. A kernel used in our simulation is the Gaussian kernel:

$$K(x, y) := e^{-\|x-y\|^2/\sigma},$$

where $\|.\|$ is the Euclidean metric in $\mathbb{R}^d$, and $\sigma > 0$ is a parameter for the function class.

Let $\mathcal{G} := \mathcal{H}$, and let the complexity measure be $I(g) = \|g\|_{\mathcal{H}}$. Thus, Eq. (6) becomes:

$$\min_w J := \min_w \frac{1}{n} \sum_{i=1}^{n} \langle w, \Phi(x_i) \rangle - \frac{1}{n} \sum_{j=1}^{n} \log\langle w, \Phi(y_j) \rangle + \frac{\lambda_n}{2} \|w\|_{\mathcal{H}}^2, \qquad (12)$$

where $\{x_i\}$ and $\{y_j\}$ are realizations of empirical data drawn from $\mathbb{Q}$ and $\mathbb{P}$, respectively. The log function is extended take value $-\infty$ for negative arguments.

**Lemma 3.** $\min_w J$ *has the following dual form:*

$$-\min_{\alpha > 0} \sum_{j=1}^{n} -\frac{1}{n} - \frac{1}{n} \log n\alpha_j + \frac{1}{2\lambda_n} \sum_{i,j} \alpha_i \alpha_j K(y_i, y_j) + \frac{1}{2\lambda_n n^2} \sum_{i,j} K(x_i, x_j) - \frac{1}{\lambda_n n} \sum_{i,j} \alpha_j K(x_i, y_j).$$

*Proof.* Let $\psi_i(w) := \frac{1}{n}\langle w, \Phi(x_i) \rangle$, $\varphi_j(w) := -\frac{1}{n}\log\langle w, \Phi(y_j) \rangle$, and $\Omega(w) = \frac{\lambda_n}{2}\|w\|_{\mathcal{H}}^2$. We have

$$
\begin{aligned}
\min_w J &= -\max_w(\langle 0, w \rangle - J(w)) = -J^*(0) \\
&= -\min_{u_i, v_j} \sum_{i=1}^{n} \psi_i^*(u_i) + \sum_{j=1}^{n} \varphi_j^*(v_j) + \Omega^*(-\sum_{i=1}^{n} u_i - \sum_{j=1}^{n} v_j),
\end{aligned}
$$

where the last line is due to the inf-convolution theorem [16]. Simple calculations yield:

$$
\begin{aligned}
\varphi_j^*(v) &= -\frac{1}{n} - \frac{1}{n}\log n\alpha_j \text{ if } v = -\alpha_j \Phi(y_j) \text{ and } +\infty \text{ otherwise} \\
\psi_i^*(u) &= 0 \text{ if } u = \frac{1}{n}\Phi(x_i) \text{ and } +\infty \text{ otherwise} \\
\Omega^*(v) &= \frac{1}{2\lambda_n}\|v\|_{\mathcal{H}}^2.
\end{aligned}
$$

So, $\min_w J = -\min_{\alpha_i} \sum_{j=1}^{n}(-\frac{1}{n} - \frac{1}{n}\log n\alpha_j) + \frac{1}{2\lambda_n}\|\sum_{j=1}^{n}\alpha_j\Phi(y_j) - \frac{1}{n}\sum_{i=1}^{n}\Phi(x_i)\|_{\mathcal{H}}^2$, which implies the lemma immediately. □

If $\hat{\alpha}$ is solution of the dual formulation, it is not difficult to show that the optimal $\hat{w}$ is attained at $\hat{w} = \frac{1}{\lambda_n}(\sum_{j=1}^{n}\hat{\alpha}_j\Phi(y_j) - \frac{1}{n}\sum_{i=1}^{n}\Phi(x_i))$.

For an RKHS based on a Gaussian kernel, the entropy condition (10) holds for any $\gamma > 0$ [23]. Furthermore, (9) trivially holds via the Cauchy-Schwarz inequality: $|g(x)| = |\langle w, \Phi(x) \rangle| \leq \|w\|_{\mathcal{H}}\|\Phi(x)\|_{\mathcal{H}} \leq I(g)\sqrt{K(x,x)} \leq I(g)$. Thus, by Theorem 2(a), $\|\hat{w}\|_{\mathcal{H}} = \|\hat{g}_n\|_{\mathcal{H}} = O_\mathbb{P}(\|g_0\|_{\mathcal{H}})$, so the penalty term $\lambda_n\|\hat{w}\|^2$ vanishes at the same rate as $\lambda_n$. We have arrived at the following estimator for the KL divergence:

$$\hat{D}_K = 1 + \sum_{j=1}^{n}(-\frac{1}{n} - \frac{1}{n}\log n\hat{\alpha}_j) = \sum_{j=1}^{n} -\frac{1}{n}\log n\hat{\alpha}_j.$$

$\log \mathcal{G}$ **is an RKHS.** Alternatively, we could set $\log \mathcal{G}$ to be the RKHS, letting $g(x) = \exp\langle w, \Phi(x) \rangle$, and letting $I(g) = \|\log g\|_{\mathcal{H}} = \|w\|_{\mathcal{H}}$. Theorem 2 is not applicable in this case, because condition (9) no longer holds, but this choice nonetheless seems reasonable and worth investigating, because in effect we have a far richer function class which might improve the bias of our estimator when the true density ratio is not very smooth.

A derivation similar to the previous case yields the following convex program:

$$
\min_w J \quad := \quad \min_w \frac{1}{n}\sum_{i=1}^n e^{\langle w,\,\Phi(x_i)\rangle} - \frac{1}{n}\sum_{j=1}^n \langle w,\,\Phi(y_j)\rangle + \frac{\lambda_n}{2}\|w\|_{\mathcal{H}}^2
$$

$$
= \quad -\min_{\alpha>0} \sum_{i=1}^n \alpha_i \log(n\alpha_i) - \alpha_i + \frac{1}{2\lambda_n}\|\sum_{i=1}^n \alpha_i \Phi(x_i) - \frac{1}{n}\sum_{j=1}^n \Phi(y_j)\|_{\mathcal{H}}^2.
$$

Letting $\hat{\alpha}$ be the solution of the above convex program, the KL divergence can be estimated by:

$$
\hat{D}_K = 1 + \sum_{i=1}^n \hat{\alpha}_i \log \hat{\alpha}_i + \hat{\alpha}_i \log \frac{n}{e}.
$$

## 5   Proof of Theorem 2

We now sketch out the proof of the main theorem. The key to our analysis is the following lemma:

**Lemma 4.** *If $\hat{g}_n$ is an estimate of $g$ using* (6)*, then:*

$$
\frac{1}{4}h_{\mathbb{Q}}^2(g_0,\hat{g}_n) + \frac{\lambda_n}{2}I^2(\hat{g}_n) \le -\int (\hat{g}_n - g_0)d(\mathbb{Q}_n - \mathbb{Q}) + \int 2\log\frac{\hat{g}_n + g_0}{2g_0}d(\mathbb{P}_n - \mathbb{P}) + \frac{\lambda_n}{2}I^2(g_0).
$$

*Proof.* Define $d_l(g_0,g) = \int (g - g_0)d\mathbb{Q} - \log\frac{g}{g_0}d\mathbb{P}$. Note that for $x > 0$, $\frac{1}{2}\log x \le \sqrt{x} - 1$. Thus, $\int \log\frac{g}{g_0}\,d\mathbb{P} \le 2\int (g^{1/2}g_0^{-1/2} - 1)\,d\mathbb{P}$. As a result, for any $g$, $d_l$ is related to $h_{\mathbb{Q}}$ as follows:

$$
\begin{aligned}
d_l(g_0,g) &\ge \int (g - g_0)\,d\mathbb{Q} - 2\int (g^{1/2}g_0^{-1/2} - 1)\,d\mathbb{P} \\
&= \int (g - g_0)\,d\mathbb{Q} - 2\int (g^{1/2}g_0^{1/2} - g_0)\,d\mathbb{Q} = \int (g^{1/2} - g_0^{1/2})^2 d\mathbb{Q} = 2h_{\mathbb{Q}}^2(g_0,g).
\end{aligned}
$$

By the definition (6) of our estimator, we have:

$$
\int \hat{g}_n d\mathbb{Q}_n - \int \log \hat{g}_n d\mathbb{P}_n + \frac{\lambda_n}{2}I^2(\hat{g}_n) \le \int g_0 d\mathbb{Q}_n - \int \log g_0 d\mathbb{P}_n + \frac{\lambda_n}{2}I^2(g_0).
$$

Both sides (modulo the regularization term $I^2$) are convex functionals of $g$. By Jensen's inequality, if $F$ is a convex function, then $F((u+v)/2) - F(v) \le (F(u) - F(v))/2$. We obtain:

$$
\int \frac{\hat{g}_n + g_0}{2}d\mathbb{Q}_n - \int \log\frac{\hat{g}_n + g_0}{2}d\mathbb{P}_n + \frac{\lambda_n}{4}I^2(\hat{g}_n) \le \int g_0 d\mathbb{Q}_n - \int \log g_0 d\mathbb{P}_n + \frac{\lambda_n}{4}I^2(g_0).
$$

Rearranging, $\int \frac{\hat{g}_n - g_0}{2}d(\mathbb{Q}_n - \mathbb{Q}) - \int \log\frac{\hat{g}_n + g_0}{2g_0}d(\mathbb{P}_n - \mathbb{P}) + \frac{\lambda_n}{4}I^2(\hat{g}_n) \le$

$$
\begin{aligned}
\int \log\frac{\hat{g}_n + g_0}{2g_0}d\mathbb{P} - \int \frac{\hat{g}_n - g_0}{2}d\mathbb{Q} + \frac{\lambda_n}{4}I^2(g_0) &= -d_l(g_0,\frac{g_0 + \hat{g}_n}{2}) + \frac{\lambda_n}{4}I^2(g_0) \\
&\le -2h_{\mathbb{Q}}^2(g_0,\frac{g_0 + \hat{g}_n}{2}) + \frac{\lambda_n}{4}I^2(g_0) \le -\frac{1}{8}h_{\mathbb{Q}}^2(g_0,\hat{g}_n) + \frac{\lambda_n}{4}I^2(g_0),
\end{aligned}
$$

where the last inequality is a standard result for the (generalized) Hellinger distance (cf. [20]). $\square$

Let us now proceed to part (a) of the theorem. Define $f_g := \log\frac{g + g_0}{2g_0}$, and let $\mathcal{F}_M := \{f_g | g \in \mathcal{G}_M\}$. Since $f_g$ is a Lipschitz function of $g$, conditions (8) and (10) imply that

$$
\mathcal{H}_\delta^B(\mathcal{F}_M, L_2(\mathbb{P})) = O(M/\delta)^\gamma. \tag{13}
$$

Apply Lemma 5.14 of [20] using distance metric $d_2(g_0,g) = \|g - g_0\|_{L_2(\mathbb{Q})}$, the following is true under $\mathbb{Q}$ (and so true under $\mathbb{P}$ as well, since $d\mathbb{P}/d\mathbb{Q}$ is bounded from above),

$$
\sup_{g\in\mathcal{G}} \frac{|\int (g - g_0)d(\mathbb{Q}_n - \mathbb{Q})|}{n^{-1/2}d_2(g_0,g)^{1-\gamma/2}(1 + I(g) + I(g_0))^{\gamma/2} \vee n^{-\frac{2}{2+\gamma}}(1 + I(g) + I(g_0))} = O_{\mathbb{P}}(1). \tag{14}
$$

In the same vein, we obtain that under $\mathbb{P}$ measure:

$$\sup_{g \in \mathcal{G}} \frac{|\int f_g d(\mathbb{P}_n - \mathbb{P})|}{n^{-1/2} d_2(g_0, g)^{1-\gamma/2} (1 + I(g) + I(g_0))^{\gamma/2} \vee n^{-\frac{2}{2+\gamma}} (1 + I(g) + I(g_0))} = O_{\mathbb{P}}(1). \quad (15)$$

By condition (9), we have: $d_2(g_0, g) = \|g - g_0\|_{L_2(\mathbb{Q})} \leq 2c^{1/2}(1 + I(g) + I(g_0))^{1/2} h_{\mathbb{Q}}(g_0, g)$. Combining Lemma 4 and Eqs. (15), (14), we obtain the following:

$$\frac{1}{4} h_{\mathbb{Q}}^2(g_0, \hat{g}_n) + \frac{\lambda_n}{2} I^2(\hat{g}_n) \leq \lambda_n I(g_0)^2/2 +$$

$$O_{\mathbb{P}}\left( n^{-1/2} h_{\mathbb{Q}}(g_0, g)^{1-\gamma/2}(1 + I(g) + I(g_0))^{1/2+\gamma/4} \vee n^{-\frac{2}{2+\gamma}}(1 + I(g) + I(g_0)) \right). \quad (16)$$

From this point, the proof involves simple algebraic manipulation of (16). To simplify notation, let $\hat{h} = h_{\mathbb{Q}}(g_0, \hat{g}_n)$, $\hat{I} = I(\hat{g}_n)$, and $I_0 = I(g_0)$. There are four possibilities:

**Case a.** $\hat{h} \geq n^{-1/(2+\gamma)}(1 + \hat{I} + I_0)^{1/2}$ and $\hat{I} \geq 1 + I_0$. From (16), either

$$\hat{h}^2/4 + \lambda_n \hat{I}^2/2 \leq O_{\mathbb{P}}(n^{-1/2})\hat{h}^{1-\gamma/2}\hat{I}^{1/2+\gamma/4} \text{ or } \hat{h}^2/4 + \lambda_n \hat{I}^2/2 \leq \lambda_n I_0^2/2,$$

which implies, respectively, either

$$\hat{h} \leq \lambda_n^{-1/2} O_{\mathbb{P}}(n^{-2/(2+\gamma)}), \quad \hat{I} \leq \lambda_n^{-1} O_{\mathbb{P}}(n^{-2/(2+\gamma)}) \text{ or}$$

$$\hat{h} \leq O_{\mathbb{P}}(\lambda_n^{1/2} I_0), \quad \hat{I} \leq O_{\mathbb{P}}(I_0).$$

Both scenarios conclude the proof if we set $\lambda_n^{-1} = O_{\mathbb{P}}(n^{2/(\gamma+2)}(1 + I_0))$.

**Case b.** $\hat{h} \geq n^{-1/(2+\gamma)}(1 + \hat{I} + I_0)^{1/2}$ and $\hat{I} < 1 + I_0$. From (16), either

$$\hat{h}^2/4 + \lambda_n \hat{I}^2/2 \leq O_{\mathbb{P}}(n^{-1/2})\hat{h}^{1-\gamma/2}(1 + I_0)^{1/2+\gamma/4} \text{ or } \hat{h}^2/4 + \lambda_n \hat{I}^2/2 \leq \lambda_n I_0^2/2,$$

which implies, respectively, either

$$\hat{h} \leq (1 + I_0)^{1/2} O_{\mathbb{P}}(n^{-1/(\gamma+2)}), \quad \hat{I} \leq 1 + I_0 \text{ or}$$

$$\hat{h} \leq O_{\mathbb{P}}(\lambda_n^{1/2} I_0), \quad \hat{I} \leq O_{\mathbb{P}}(I_0).$$

Both scenarios conclude the proof if we set $\lambda_n^{-1} = O_{\mathbb{P}}(n^{2/(\gamma+2)}(1 + I_0))$.

**Case c.** $\hat{h} \leq n^{-1/(2+\gamma)}(1 + \hat{I} + I_0)^{1/2}$ and $\hat{I} \geq 1 + I_0$. From (16)

$$\hat{h}^2/4 + \lambda_n \hat{I}^2/2 \leq O_{\mathbb{P}}(n^{-2/(2+\gamma)})\hat{I},$$

which implies that $\hat{h} \leq O_{\mathbb{P}}(n^{-1/(2+\gamma)})\hat{I}^{1/2}$ and $\hat{I} \leq \lambda_n^{-1} O_{\mathbb{P}}(n^{-2/(2+\gamma)})$. This means that $\hat{h} \leq O_{\mathbb{P}}(\lambda_n^{1/2})(1 + I_0)$, $\hat{I} \leq O_{\mathbb{P}}(1 + I_0)$ if we set $\lambda_n^{-1} = O_{\mathbb{P}}(n^{2/(2+\gamma)})(1 + I_0)$.

**Case d.** $\hat{h} \leq n^{-1/(2+\gamma)}(1 + \hat{I} + I_0)^{1/2}$ and $\hat{I} \leq 1 + I_0$. Part (a) of the theorem is immediate.

Finally, part (b) is a simple consequence of part (a) using the same argument as in Thm. 9 of [15].

## 6   Simulation results

In this section, we describe the results of various simulations that demonstrate the practical viability of our estimators, as well as their convergence behavior. We experimented with our estimators using various choices of $\mathbb{P}$ and $\mathbb{Q}$, including Gaussian, beta, mixture of Gaussians, and multivariate Gaussian distributions. Here we report results in terms of KL estimation error. For each of the eight estimation problems described here, we experiment with increasing sample sizes (the sample size, $n$, ranges from 100 to $10^4$ or more). Error bars are obtained by replicating each set-up 250 times.

For all simulations, we report our estimator's performance using the simple fixed rate $\lambda_n \sim 1/n$, noting that this may be a suboptimal rate. We set the kernel width to be relatively small ($\sigma = .1$) for one-dimension data, and larger for higher dimensions. We use M1 to denote the method in which $\mathcal{G}$ is the RKHS, and M2 for the method in which $\log \mathcal{G}$ is the RKHS. Our methods are compared to

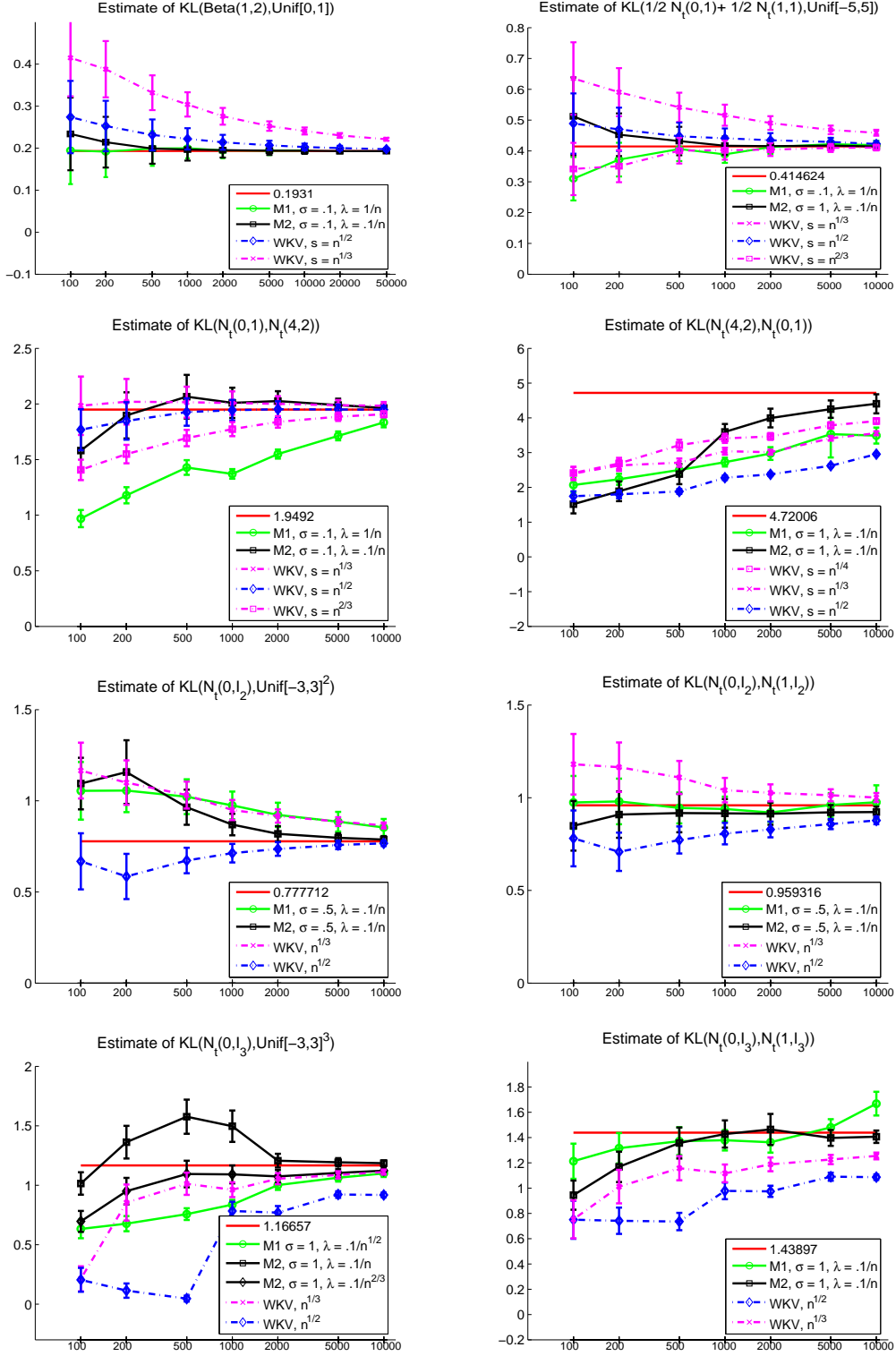

**Figure 1.** Results of estimating KL divergences for various choices of probability distributions. In all plots, the X-axis is the number of data points plotted on a log scale, and the Y-axis is the estimated value. The error bar is obtained by replicating the experiment 250 times. $N_t(a, I_k)$ denotes a truncated normal distribution of $k$ dimensions with mean $(a, \ldots, a)$ and identity covariance matrix.

algorithm $A$ in Wang et al [22], which was shown empirically to be one of the best methods in the literature. Their method, denoted by WKV, is based on data-dependent partitioning of the covariate space. Naturally, the performance of WKV is critically dependent on the amount $s$ of data allocated to each partition; here we report results with $s \sim n^\gamma$, where $\gamma = 1/3, 1/2, 2/3$.

The first four plots present results with univariate distributions. In the first two, our estimators $M1$ and $M2$ appear to have faster convergence rate than WKV. The WKV estimator performs very well in the third example, but rather badly in the fourth example. The next four plots present results with two and three dimensional data. Again, M1 has the best convergence rates in all examples. The M2 estimator does not converge in the last example, suggesting that the underlying function class exhibits very strong bias. The WKV methods have weak convergence rates despite different choices of the partition sizes. It is worth noting that as one increases the number of dimensions, histogram based methods such as WKV become increasingly difficult to implement, whereas increasing dimension has only a mild effect on our method.

## References

[1] S. M. Ali and S. D. Silvey. A general class of coefficients of divergence of one distribution from another. *J. Royal Stat. Soc. Series B*, 28:131–142, 1966.

[2] P. L. Bartlett, M. I. Jordan, and J. D. McAuliffe. Convexity, classification, and risk bounds. *Journal of the American Statistical Association*, 101:138–156, 2006.

[3] P. Bickel and Y. Ritov. Estimating integrated squared density derivatives: Sharp best order of convergence estimates. *Sankhyā Ser. A*, 50:381–393, 1988.

[4] L. Birgé and P. Massart. Estimation of integral functionals of a density. *Ann. Statist.*, 23(1):11–29, 1995.

[5] M. Broniatowski and A. Keziou. Parametric estimation and tests through divergences. Technical report, LSTA, Université Pierre et Marie Curie, 2004.

[6] I. Csiszár. Information-type measures of difference of probability distributions and indirect observation. *Studia Sci. Math. Hungar*, 2:299–318, 1967.

[7] L. Gyorfi and E.C. van der Meulen. Density-free convergence properties of various estimators of entropy. *Computational Statistics and Data Analysis*, 5:425–436, 1987.

[8] P. Hall and S. Morton. On estimation of entropy. *Ann. Inst. Statist. Math.*, 45(1):69–88, 1993.

[9] I. A. Ibragimov and R. Z. Khasminskii. On the nonparametric estimation of functionals. In *Symposium in Asymptotic Statistics*, pages 41–52, 1978.

[10] H. Joe. Estimation of entropy and other functionals of a multivariate density. *Ann. Inst. Statist. Math.*, 41:683–697, 1989.

[11] A. Keziou. Dual representation of $\phi$-divergences and applications. *C. R. Acad. Sci. Paris, Ser. I 336*, pages 857–862, 2003.

[12] B. Laurent. Efficient estimation of integral functionals of a density. *Ann. Statist.*, 24(2):659–681, 1996.

[13] B. Ya. Levit. Asymptotically efficient estimation of nonlinear functionals. *Problems Inform. Transmission*, 14:204–209, 1978.

[14] X. Nguyen, M. J. Wainwright, and M. I. Jordan. On divergences, surrogate losses and decentralized detection. Technical Report 695, Dept of Statistics, UC Berkeley, October 2005.

[15] X. Nguyen, M. J. Wainwright, and M. I. Jordan. Nonparametric estimation of the likelihood ratio and divergence functionals. In *International Symposium on Information Theory (ISIT)*, 2007.

[16] G. Rockafellar. *Convex Analysis*. Princeton University Press, Princeton, 1970.

[17] S. Saitoh. *Theory of Reproducing Kernels and its Applications*. Longman, Harlow, UK, 1988.

[18] B. Schölkopf and A. Smola. *Learning with Kernels*. MIT Press, Cambridge, MA, 2002.

[19] F. Topsoe. Some inequalities for information divergence and related measures of discrimination. *IEEE Transactions on Information Theory*, 46:1602–1609, 2000.

[20] S. van de Geer. *Empirical Processes in M-Estimation*. Cambridge University Press, 2000.

[21] A. W. van der Vaart and J. Wellner. *Weak Convergence and Empirical Processes*. Springer-Verlag, New York, NY, 1996.

[22] Q. Wang, S. R. Kulkarni, and S. Verdú. Divergence estimation of continuous distributions based on data-dependent partitions. *IEEE Transactions on Information Theory*, 51(9):3064–3074, 2005.

[23] D. X. Zhou. The covering number in learning theory. *Journal of Complexity*, 18:739–767, 2002.

